# Monotonicity Hints

**Joseph Sill**
Computation and Neural Systems program
California Institute of Technology
email: joe@cs.caltech.edu

**Yaser S. Abu-Mostafa**
EE and CS Deptartments
California Institute of Technology
email: yaser@cs.caltech.edu

## Abstract

A hint is any piece of side information about the target function to be learned. We consider the monotonicity hint, which states that the function to be learned is monotonic in some or all of the input variables. The application of monotonicity hints is demonstrated on two real-world problems- a credit card application task, and a problem in medical diagnosis. A measure of the monotonicity error of a candidate function is defined and an objective function for the enforcement of monotonicity is derived from Bayesian principles. We report experimental results which show that using monotonicity hints leads to a statistically significant improvement in performance on both problems.

## 1 Introduction

Researchers in pattern recognition, statistics, and machine learning often draw a contrast between linear models and nonlinear models such as neural networks. Linear models make very strong assumptions about the function to be modelled, whereas neural networks are said to make no such assumptions and can in principle approximate any smooth function given enough hidden units. Between these two extremes, there exists a frequently neglected middle ground of nonlinear models which incorporate strong prior information and obey powerful constraints.

A monotonic model is one example which might occupy this middle area. Monotonic models would be more flexible than linear models but still highly constrained. Many applications arise in which there is good reason to believe the target function is monotonic in some or all input variables. In screening credit card applicants, for instance, one would expect that the probability of default decreases monotonically

with the applicant's salary. It would be very useful, therefore, to be able to constrain a nonlinear model to obey monotonicity.

The general framework for incorporating prior information into learning is well established and is known as learning from hints[1]. A hint is any piece of information about the target function beyond the available input-output examples. Hints can improve the performance of learning models by reducing capacity without sacrificing approximation ability [2]. Invariances in character recognition [3] and symmetries in financial-market forecasting [4] are some of the hints which have proven beneficial in real-world learning applications. This paper describes the first practical applications of monotonicity hints. The method is tested on two noisy real-world problems: a classification task concerned with credit card applications and a regression problem in medical diagnosis.

Section II derives, from Bayesian principles, an appropriate objective function for simultaneously enforcing monotonicity and fitting the data. Section III describes the details and results of the experiments. Section IV analyzes the results and discusses possible future work.

## 2 Bayesian Interpretation of Objective Function

Let $\mathbf{x}$ be a vector drawn from the input distribution and $\mathbf{x}'$ be such that

$$\forall j \neq i, x'_j = x_j \tag{1}$$

$$x'_i > x_i \tag{2}$$

The statement that $f$ is monotonically increasing in input variable $x_i$ means that for all such $\mathbf{x}, \mathbf{x}'$ defined as above

$$f(\mathbf{x}') \geq f(\mathbf{x}) \tag{3}$$

Decreasing monotonicity is defined similarly.

We wish to define a single scalar measure of the degree to which a particular candidate function $y$ obeys monotonicity in a set of input variables.

One such natural measure, the one used in the experiments in Section IV, is defined in the following way: Let $\mathbf{x}$ be an input vector drawn from the input distribution. Let $i$ be the index of an input variable randomly chosen from a uniform distribution over those variables for which monotonicity holds. Define a perturbation distribution, e.g., U[0,1], and draw $\delta x_i$ from this distribution. Define $\mathbf{x}'$ such that

$$\forall j \neq i, x'_j = x_j \tag{4}$$

$$x'_i = x_i + sgn(i)\delta x_i \tag{5}$$

where $sgn(i) = 1$ or $-1$ depending on whether $f$ is monotonically increasing or decreasing in variable i. We will call $E_h$ the *monotonicity error* of $y$ on the input pair $(\mathbf{x}, \mathbf{x}')$.

$$E_h = \begin{cases} 0 & y(\mathbf{x}') \geq y(\mathbf{x}) \\ (y(\mathbf{x}) - y(\mathbf{x}'))^2 & y(\mathbf{x}') < y(\mathbf{x}) \end{cases} \tag{6}$$

Our measure of $y$'s violation of monotonicity is $\mathcal{E}[E_h]$, where the expectation is taken with respect to random variables $\mathbf{x}$, $i$ and $\delta x_i$.

We believe that the best possible approximation to $f$ given the architecture used is probably approximately monotonic. This belief may be quantified in a prior distribution over the candidate functions implementable by the architecture:

$$Pr(y) \propto e^{-\lambda \mathcal{E}[E_h]} \tag{7}$$

This distribution represents the *a priori* probability density, or likelihood, assigned to a candidate function with a given level of monotonicity error. The probability that a function is the best possible approximation to $f$ decreases exponentially with the increase in monotonicity error. $\lambda$ is a positive constant which indicates how strong our bias is towards monotonic functions.

In addition to obeying prior information, the model should fit the data well. For classification problems, we take the network output $y$ to represent the probability of class $c = 1$ conditioned on the observation of the input vector (the two possible classes are denoted by 0 and 1). We wish to pick the most probable model given the data. Equivalently, we may choose to maximize $log(P(model|data))$. Using Bayes' Theorem,

$$log(P(model|data)) \propto log(P(data|model) + log(P(model)) \tag{8}$$

$$= \sum_{m=1}^{M} c_m log(y_m) + (1 - c_m)log(1 - y_m) - \lambda \mathcal{E}[E_h] \tag{9}$$

For continuous-output regression problems, we interpret y as the conditional mean of the observed output $t$ given the observation of $\mathbf{x}$. If we assume constant-variance gaussian noise, then by the same reasoning as in the classification case, the objective function to be maximized is :

$$-\sum_{m=1}^{M} (y_m - t_m)^2 - \lambda \mathcal{E}[E_h] \tag{10}$$

The Bayesian prior leads to a familiar form of objective function, with the first term reflecting the desire to fit the data and a second term penalizing deviation from monotonicity.

## 3  Experimental Results

Both databases were obtained via FTP from the machine learning database repository maintained by UC-Irvine [1].

The credit card task is to predict whether or not an applicant will default. For each of 690 applicant case histories, the database contains 15 features describing the applicant plus the class label indicating whether or not a default ultimately occurred. The meaning of the features is confidential for proprietary reasons. Only the 6 continuous features were used in the experiments reported here. 24 of the case histories had at least one feature missing. These examples were omitted, leaving 666 which were used in the experiments. The two classes occur with almost equal frequency; the split is 55%-45%.

Intuition suggests that the classification should be monotonic in the features. Although the specific meanings of the continuous features are not known, we assume here that they represent various quantities such as salary, assets, debt, number of years at current job, etc. Common sense dictates that the higher the salary or the lower the debt, the less likely a default is, all else being equal. Monotonicity in all features was therefore asserted.

The motivation in the medical diagnosis problem is to determine the extent to which various blood tests are sensitive to disorders related to excessive drinking. Specifically, the task is to predict the number of drinks a particular patient consumes per day given the results of 5 blood tests. 345 patient histories were collected, each consisting of the 5 test results and the daily number of drinks. The "number of drinks" variable was normalized to have variance 1. This normalization makes the results easier to interpret, since a trivial mean-squared-error performance of 1.0 may be obtained by simply predicting for mean number of drinks for each patient, irrespective of the blood tests.

The justification for monotonicity in this case is based on the idea that an abnormal result for each test is indicative of excessive drinking, where abnormal means either abnormally high or abnormally low.

In all experiments, batch-mode backpropagation with a simple adaptive learning rate scheme was used [2]. Several methods were tested. The performance of a linear perceptron was observed for benchmark purposes. For the experiments using nonlinear methods, a single hidden layer neural network with 6 hidden units and direct input-output connections was used on the credit data; 3 hidden units and direct input-output connections were used for the liver task. The most basic method tested was simply to train the network on all the training data and optimize the objective function as much as possible. Another technique tried was to use a validation set to avoid overfitting. Training for all of the above models was performed by maximizing only the first term in the objective function, i.e., by maximizing the log-likelihood of the data (minimizing training error). Finally, training the networks with the monotonicity constraints was performed, using an approximation to (9)

and (10).

A leave-k-out procedure was used in order to get statistically significant comparisons of the difference in performance. For each method, the data was randomly partitioned 200 different ways (The split was 550 training, 116 test for the credit data; 270 training and 75 test for the liver data). The results shown in Table 1 are averages over the 200 different partitions.

In the early stopping experiments, the training set was further subdivided into a set (450 for the credit data, 200 for the liver data) used for direct training and a second validation set (100 for the credit data, 70 for the liver data). The classification error on the validation set was monitored over the entire course of training, and the values of the network weights at the point of lowest validation error were chosen as the final values.

The process of training the networks with the monotonicity hints was divided into two stages. Since the meanings of the features were unaccessible, the directions of monotonicity were not known *a priori*. These directions were determined by training a linear perceptron on the training data for 300 iterations and observing the resulting weights. A positive weight was taken to imply increasing monotonicity, while a negative weight meant decreasing monotonicity.

Once the directions of monotonicity were determined, the networks were trained with the monotonicity hints. For the credit problem, an approximation to the theoretical objective function (10) was maximized:

$$\frac{1}{M} \sum_{m=1}^{M} c_m log(y_m) + (1 - c_m) log(1 - y_m) - \frac{\lambda}{N} \sum_{n=1}^{N} E_{h,n} \qquad (13)$$

For the liver problem, objective function (12) was approximated by

$$-\frac{1}{M} \sum_{m=1}^{M} (y_m - t_m)^2 - \frac{\lambda}{N} \sum_{n=1}^{N} E_{h,n} \qquad (14)$$

$E_{h,n}$ represents the network's monotonicity error on a particular pair of input vectors $\mathbf{x}, \mathbf{x}'$. Each pair was generated according to the method described in Section II. The input distribution was modelled as a joint gaussian with a covariance matrix estimated from the training data.

For each input variable, 500 pairs of vectors representing monotonicity in that variable were generated. This yielded a total of N=3000 hint example pairs for the credit problem and N=2500 pairs for the liver problem. $\lambda$ was chosen to be 5000. No optimization of $\lambda$ was attempted; 5000 was chosen somewhat arbitrarily as simply a high value which would greatly penalize non-monotonicity. Hint generalization, i.e. monotonicity test error, was measured by using 100 pairs of vectors for each variable which were not trained on but whose monotonicity error was calculated. For contrast, monotonicity test error was also monitored for the two-layer networks trained only on the input-output examples. Figure 1 shows test error and monotonicity error vs. training time for the credit data for the networks trained only on the training data (i.e, no hints), averaged over the 200 different data splits.

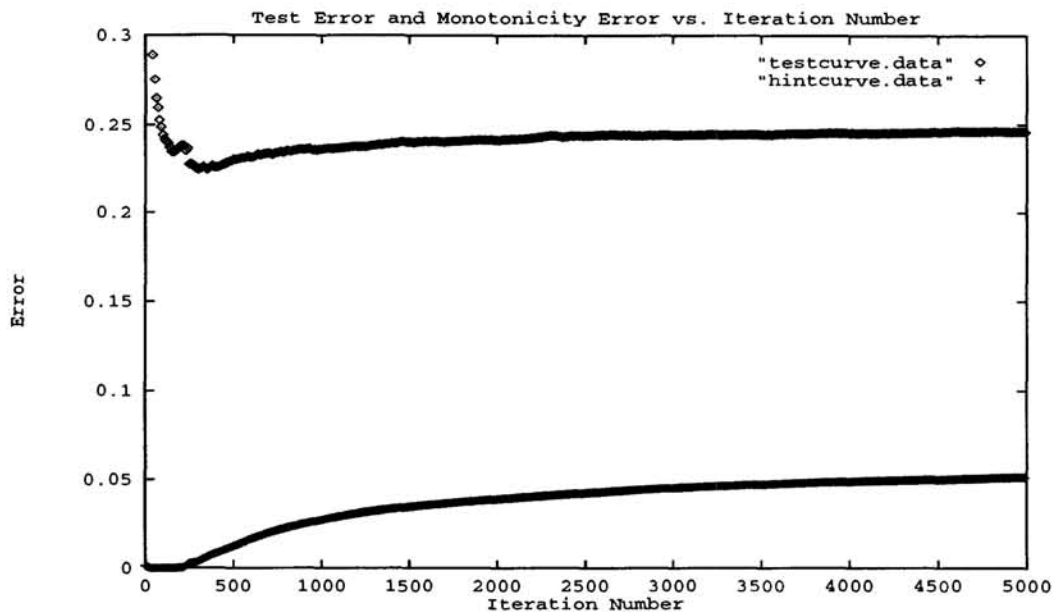

Figure 1: The violation of monotonicity tracks the overfitting occurring during training

The monotonicity error is multiplied by a factor of 10 in the figure to make it more easily visible. The figure indicates a substantial correlation between overfitting and monotonicity error during the course of training. The curves for the liver data look similar but are omitted due to space considerations.

| Method | training error | test error | hint test error |
|---|---|---|---|
| Linear | $22.7\% \pm 0.1\%$ | $23.7\% \pm 0.2\%$ | - |
| 6-6-1 net | $15.2\% \pm 0.1\%$ | $24.6\% \pm 0.3\%$ | .005115 |
| 6-6-1 net, w/val. | $18.8\% \pm 0.2\%$ | $23.4\% \pm 0.3\%$ | - |
| 6-6-1 net, w/hint | $18.7\% \pm 0.1\%$ | $21.8\% \pm 0.2\%$ | .000020 |

Table 1: Performance of methods on credit problem

The performance of each method is shown in tables 1 and 2. Without early stopping, the two-layer network overfits and performs worse than a linear model. Even with early stopping, the performance of the linear model and the two-layer network are almost the same; the difference is not statistically significant. This similarity in performance is consistent with the thesis of a monotonic target function. A monotonic classifier may be thought of as a mildly nonlinear generalization of a linear classifier. The two-layer network does have the advantage of being able to implement some of this nonlinearity. However, this advantage is cancelled out (and in other cases could be outweighed) by the overfitting resulting from excessive and unnecessary degrees of freedom. When monotonicity hints are introduced, much of this unnecessary freedom is eliminated, although the network is still allowed to implement monotonic nonlinearities. Accordingly, a modest but clearly statistically significant improvement on the credit problem (nearly 2%) results from the introduction of

| Method | training error | test error | hint test error |
|---|---|---|---|
| Linear | .802 ± .005 | .873 ± .013 | - |
| 5-3-1 net | .640 ± .003 | .920 ± .014 | .004967 |
| 5-3-1 net, w/val. | .758 ± .008 | .871 ± .013 | - |
| 5-3-1 net, w/hint | .758 ± .003 | .830 ± .013 | .000002 |

Table 2: Performance of methods on liver problem

monotonicity hints. Such an improvement could translate into a substantial increase in profit for a bank. Monotonicity hints also significantly improve test error on the liver problem; 4% more of the target variance is explained.

## 4   Conclusion

This paper has shown that monotonicity hints can significantly improve the performance of a neural network on two noisy real-world tasks. It is worthwhile to note that the beneficial effect of imposing monotonicity does not necessarily imply that the target function is entirely monotonic. If there exist some non-monotonicities in the target function, then monotonicity hints may result in some decrease in the model's ability to implement this function. It may be, though, that this penalty is outweighed by the improved estimation of model parameters due to the decrease in model complexity. Therefore, the use of monotonicity hints probably should be considered in cases where the target function is thought to be at least roughly monotonic and the training examples are limited in number and noisy.

Future work may include the application of monotonicity hints to other real world problems and further investigations into techniques for enforcing the hints.

### Acknowledgements

The authors thank Eric Bax, Zehra Cataltepe, Malik Magdon-Ismail, and Xubo Song for many useful discussions.

## Footnotes

[1] They may be obtained as follows: ftp ics.uci.edu. cd pub/machine-learning-databases. The credit data is in the subdirectory /credit-screening, while the liver data is in the subdirectory /liver-disorders.

[2] If the previous iteration resulted in a increase in likelihood, the learning rate was increased by 3%. If the likelihood decreased, the learning rate was cut in half

### References

[1] Y. Abu-Mostafa (1990). Learning from Hints in Neural Networks *Journal of Complexity* 6, 192-198.

[2] Y. Abu-Mostafa (1993) Hints and the VC Dimension *Neural Computation* 4, 278-288

[3] P. Simard, Y. LeCun & J Denker (1993) Efficient Pattern Recognition Using a New Transformation Distance *NIPS5*, 50-58.

[4] Y. Abu-Mostafa (1995) Financial Market Applications of Learning from Hints *Neural Networks in the Capital Markets*, A. Refenes, ed., 221-232. Wiley, London, UK.